# *Discovering the Structure of a Reactive Environment by Exploration*

**Michael C. Mozer**
Department of Computer Science
and Institute of Cognitive Science
University of Colorado
Boulder, CO  80309-0430

**Jonathan Bachrach**
Department of Computer
and Information Science
University of Massachusetts
Amherst, MA  01003

## ABSTRACT

Consider a robot wandering around an unfamiliar environment, performing actions and sensing the resulting environmental states. The robot's task is to construct an internal model of its environment, a model that will allow it to predict the consequences of its actions and to determine what sequences of actions to take to reach particular goal states. Rivest and Schapire (1987a, 1987b; Schapire, 1988) have studied this problem and have designed a symbolic algorithm to strategically explore and infer the structure of "finite state" environments. The heart of this algorithm is a clever representation of the environment called an *update graph*. We have developed a connectionist implementation of the update graph using a highly-specialized network architecture. With back propagation learning and a trivial exploration strategy — choosing random actions — the connectionist network can outperform the Rivest and Schapire algorithm on simple problems. The network has the additional strength that it can accommodate stochastic environments. Perhaps the greatest virtue of the connectionist approach is that it suggests generalizations of the update graph representation that do not arise from a traditional, symbolic perspective.

## 1  INTRODUCTION

Consider a robot placed in an unfamiliar environment. The robot is allowed to wander around the environment, performing actions and sensing the resulting environmental states. With sufficient exploration, the robot should be able to construct an internal model of the environment, a model that will allow it to predict the consequences of its actions and to determine what sequence of actions must be taken to reach a particular goal state. In this paper, we describe a connectionist network that accomplishes this task, based on a representation of finite-state automata developed by Rivest and Schapire

(1987a, 1987b; Schapire, 1988).

The environments we wish to consider can be modeled by a finite-state automaton (*FSA*). In each environment, the robot has a set of discrete *actions* it can execute to move from one environmental state to another. At each environmental state, a set of binary-valued *sensations* can be detected by the robot. To illustrate the concepts and methods in our work, we use as an extended example a simple environment, the $n$-room world (from Rivest and Schapire). The $n$-room world consists of $n$ rooms arranged in a circular chain. Each room is connected to the two adjacent rooms. In each room is a light bulb and light switch. The robot can sense whether the light in the room where it currently stands is on or off. The robot has three possible actions: move to the next room down the chain (D), move to the next room up the chain (U), and toggle the light switch in the current room (T).

## 2  MODELING THE ENVIRONMENT

If the FSA corresponding to the $n$-room world is known, the sensory consequences of any sequence of actions can be predicted. Further, the FSA can be used to determine a sequence of actions to take to obtain a certain goal state. Although one might try developing an algorithm to learn the FSA directly, there are several arguments against doing so (Schapire, 1988). Most important is that the FSA often does not capture structure inherent in the environment. Rather than trying to learn the FSA, Rivest and Schapire suggest learning another representation of the environment called an *update graph*. The advantage of the update graph is that in environments with regularities, the number of nodes in the update graph can be much smaller than in the FSA (e.g., $2n$ versus $2^n$ for the $n$-room world). Rivest and Schapire's formal definition of the update graph is based on the notion of *tests* that can be performed on the environment, and the equivalence of different tests. In this section, we present an alternative, more intuitive view of the update graph that facilitates a connectionist interpretation.

Consider a three-room world. To model this environment, the essential knowledge required is the status of the lights in the current room (CUR), the next room up from the current room (UP), and the next room down from the current room (DOWN). Assume the update graph has a node for each of these environmental variables. Further assume that each node has an associated value indicating whether the light in the particular room is on or off.

If we know the values of the variables in the current environmental state, what will their new values be after taking some action, say U? When the robot moves to the next room up, the new value of CUR becomes the previous value of UP; the new value of DOWN becomes the previous value of CUR; and in the three-room world, the new value of UP becomes the previous value of DOWN. As depicted in Figure 1a, this action thus results in shifting values around in the three nodes. This makes sense because moving up does not affect the status of any light, but it does alter the robot's position with respect to the three rooms. Figure 1b shows the analogous flow of information for the action D. Finally, the action T should cause the status of the current room's light to be complemented while the other two rooms remain unaffected (Figure 1c). In Figure 1d, the three sets of links from Figures 1a-c have been superimposed and have been labeled with the appropriate action.

One final detail: The Rivest and Schapire update graph formalism does not make use of the "complementation" link. To avoid it, each node may be split into two values, one

representing the status of a room and the other its complement (Figure 1e). Toggling thus involves exchanging the values of CUR and $\overline{\text{CUR}}$. Just as the values of CUR, UP, and DOWN must be shifted for the actions U and D, so must their complements.

Given the update graph in Figure 1e and the value of each node for the current environmental state, the result of any sequence of actions can be predicted simply by shifting values around in the graph. Thus, as far as predicting the input/output behavior of the environment is concerned, the update graph serves the same purpose as the FSA.

A defining and nonobvious (from the current description) property of an update graph is that each node has exactly one incoming link for each action. We call this the *one-input-per-action* constraint. For example, CUR gets input from $\overline{\text{CUR}}$ for the action T, from UP for U, and from DOWN for D.

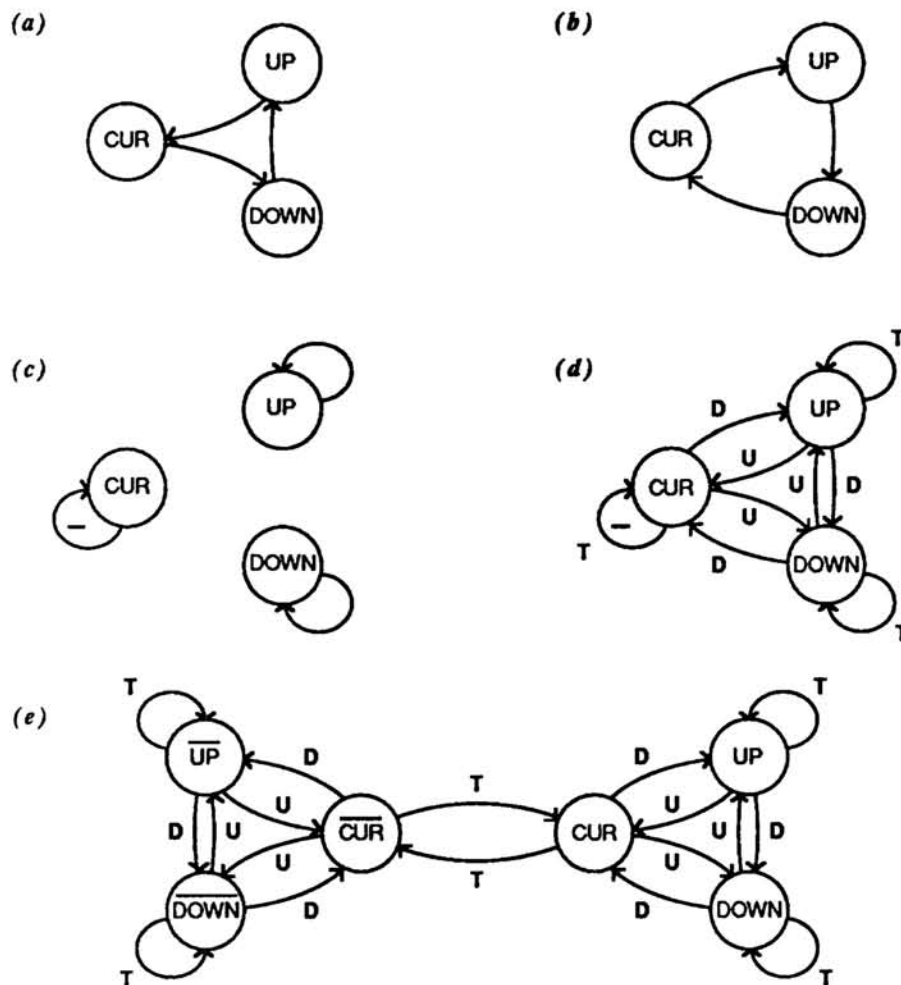

**Figure 1:** (a) Links between nodes indicating the desired information flow on performing the action U. CUR represents that status of the lights in the current room, UP the status of the lights in the next room up, and DOWN the status of the lights in the next room down. (b) Links between nodes indicating the desired information flow on performing the action D. (c) Links between nodes indicating the desired information flow on performing the action T. The "−" on the link from CUR to itself indicates that the value must be complemented. (d) Links from the three separate actions superimposed and labeled by the action. (e) The complementation link can be avoided by adding a set of nodes that represent the complements of the original set. This is the update graph for a three-room world.

## 3   THE RIVEST AND SCHAPIRE ALGORITHM

Rivest and Schapire have developed a symbolic algorithm (hereafter, *the RS algorithm*) to strategically explore an environment and learn its update graph representation. The RS algorithm formulates explicit hypotheses about regularities in the environment and tests these hypotheses one or a relatively small number at a time. As a result, the algorithm may not make full use of the environmental feedback obtained. It thus seems worthwhile to consider alternative approaches that could allow more efficient use of the environmental feedback, and hence, more efficient learning of the update graph. We have taken connectionist approach, which has shown quite promising results in preliminary experiments and suggests other significant benefits. We detail these benefits below, but must first describe the basic approach.

## 4   THE UPDATE GRAPH AS A CONNECTIONIST NETWORK

How might we turn the update graph into a connectionist network? Start by assuming one unit in a network for each node in the update graph. The activity level of the unit represents the truth value associated with the update graph node. Some of these units serve as "outputs" of the network. For example, in the three-room world, the output of the network is the unit that represents the status of the current room. In other environments, there may several sensations in which case there will be several output units.

What is the analog of the labeled links in the update graph? The labels indicate that values are to be sent down a link when a particular action occurs. In connectionist terms, the links should be *gated* by the action. To elaborate, we might include a set of units that represent the possible actions; these units act to multiplicatively gate the flow of activity between units in the update graph. Thus, when a particular action is to be performed, the corresponding action unit is activated, and the connections that are gated by this action become enabled. If the action units form a local representation, i.e., only one is active at a time, exactly one set of connections is enabled at a time. Consequently, the gated connections can be replaced by a set of weight matrices, one per action. To predict the consequences of a particular action, the weight matrix for that action is plugged into the network and activity is allowed to propagate through the connections. Thus, the network is dynamically rewired contingent on the current action.

The effect of activity propagation should be that the new activity of a unit is the previous activity of some other unit. A linear activation function is sufficient to achieve this:

$$\mathbf{X}(t) = \mathbf{W}_{a(t)}\mathbf{X}(t-1), \tag{1}$$

where $a(t)$ is the action selected at time $t$, $\mathbf{W}_{a(t)}$ is the weight matrix associated with this action, and $\mathbf{X}(t)$ is the activity vector that results from taking action $a(t)$. Assuming weight matrices which have zeroes in each row except for one connection of strength 1 (the one-input-per-action constraint), the activation rule will cause activity values to be copied around the network.

## 5   TRAINING THE NETWORK TO BE AN UPDATE GRAPH

We have described a connectionist network that can behave as an update graph, and now turn to the procedure used to learn the connection strengths in this network. For expository purposes, assume that the number of units in the update graph is known in advance.

(This is not necessary, as we show in Mozer & Bachrach, 1989.) A weight matrix is required for each action, with a potential non-zero connection between every pair of units. As in most connectionist learning procedures, the weight matrices are initialized to random values; the outcome of learning will be a set of matrices that represent the update graph connectivity.

If the network is to behave as an update graph, the one-input-per-action constraint must be satisfied. In terms of the connectivity matrices, this means that each row of each weight matrix should have connection strengths of zero except for one value which is 1. To achieve this property, additional constraints are placed on the weights. We have explored a combination of three constraints:

$$(1) \sum_j w_{aij}^2 = 1, \quad (2) \sum_j w_{aij} = 1, \quad \text{and (3) } w_{aij} \geq 0,$$

where $w_{aij}$ is the connection strength to $i$ from $j$ for action $a$. Constraint 1 is satisfied by introducing an additional cost term to the error function. Constraints 2 and 3 are rigidly enforced by renormalizing the $\mathbf{W}_{ai}$ following each weight update. The normalization procedure finds the shortest distance projection from the updated weight vector to the hyperplane specified by constraint 2 that also satisfies constraint 3.

At each time step $t$, the training procedure consists the following sequence of events:

1. An action, $a(t)$, is selected at random.

2. The weight matrix for that action, $\mathbf{W}_{a(t)}$, is used to compute the activities at $t$, $\mathbf{X}(t)$, from the previous activities $\mathbf{X}(t-1)$.

3. The selected action is performed on the environment and the resulting sensations are observed.

4. The observed sensations are compared with the sensations predicted by the network (i.e., the activities of units chosen to represent the sensations) to compute a measure of error. To this error is added the contribution of constraint 1.

5. The back propagation "unfolding-in-time" procedure (Rumelhart, Hinton, & Williams, 1986) is used to compute the derivative of the error with respect to weights at the current and earlier time steps, $\mathbf{W}_{a(t-i)}$, for $i=0 \cdots \tau-1$.

6. The weight matrices for each action are updated using the overall error gradient and then are renormalized to enforce constraints 2 and 3.

7. The temporal record of unit activities, $\mathbf{X}(t-i)$ for $i=0 \cdots \tau$, which is maintained to permit back propagation in time, is updated to reflect the new weights. (See further explanation below.)

8. The activities of the output units at time $t$, which represent the predicted sensations, are replaced by the actual observed sensations.

Steps 5-7 require further elaboration. The error measured at time $t$ may be due to incorrect propagation of activities from time $t-1$, which would call for modification of the weight matrix $\mathbf{W}_{a(t)}$. But the error may also be attributed to incorrect propagation of activities at earlier times. Thus, back propagation is used to assign blame to the weights at earlier times. One critical parameter of training is the amount of temporal history, $\tau$, to consider. We have found that, for a particular problem, error propagation beyond a cer-

tain critical number of steps does not improve learning performance, although any fewer does indeed harm performance. In the results described below, we set $\tau$ for a particular problem to what appeared to be a safe limit: one less than the number of nodes in the update graph solution of the problem.

To back propagate error in time, we maintain a temporal record of unit activities. However, a problem arises with these activities following a weight update: the activities are no longer consistent with the weights — i.e., Equation 1 is violated. Because the error derivatives computed by back propagation are exact only when Equation 1 is satisfied, future weight updates based on the inconsistent activities are not assured of being correct. Empirically, we have found the algorithm extremely unstable if we do not address this problem.

In most situations where back propagation is applied to temporally-extended sequences, the sequences are of finite length. Consequently, it is possible to wait until the end of the sequence to update the weights, at which point consistency between activities and weights no longer matters because the system starts fresh at the beginning of the next sequence. In the present situation, however, the sequence of actions does not terminate. We thus were forced to consider alternative means of ensuring consistency. The most successful approach involved updating the activities after each weight change to force consistency (step 7 of the list above). To do this, we propagated the earliest activities in the temporal record, $X(t-\tau)$, forward again to time $t$, using the updated weight matrices.

## 6   RESULTS

Figure 2 shows the weights in the update graph network for the three-room world after the robot has taken 6,000 steps. The Figure depicts a connectivity pattern identical to that of the update graph of Figure 1e. To explain the correspondence, think of the diagram as being in the shape of a person who has a head, left and right arms, left and right legs, and a heart. For the action U, the head — the output unit — receives input from the left leg, the left leg from the heart, and the heart from the head, thereby forming a three-unit loop. The other three units — the left arm, right arm, and right leg — form a

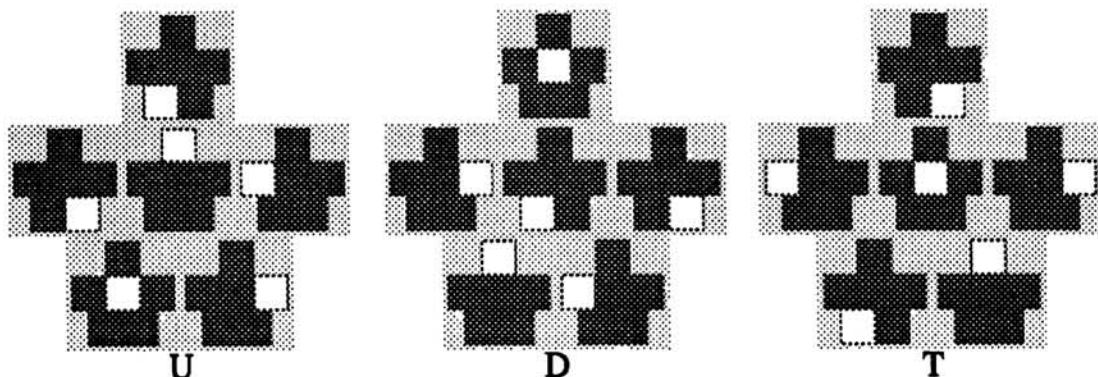

Figure 2: Weights learned after 6,000 exploratory steps in the three-room world. Each large diagram represents the weights corresponding to one of the three actions. Each small diagram contained within a large diagram represents the connection strengths feeding into a particular unit for a particular action. There are six units, hence six small diagrams. The output unit, which indicates the state of the light in the current room, is the protruding "head" of the large diagram. A white square in a particular position of a small diagram represents the strength of connection from the unit in the homologous position in the large diagram to the unit represented by the small diagram. The area of the square is proportional to the connection strength.

similar loop. For the action D, the same two loops are present but in the reverse direction. These two loops also appear in Figure 1e. For the action T, the left and right arms, heart, and left leg each keep their current value, while the head and the right leg exchange values. This corresponds to the exchange of values between the CUR and $\overline{\text{CUR}}$ nodes of the Figure 1e.

In addition to learning the update graph connectivity, the network has simultaneously learned the correct activity values associated with each node for the current state of the environment. Armed with this information, the network can predict the outcome of any sequence of actions. Indeed, the prediction error drops to zero, causing learning to cease and the network to become completely stable.

Now for the bad news: The network does not converge for every set of random initial weights, and when it does, it requires on the order of 6,000 steps. However, when the weight constraints are removed, that the network converges without fail and in about 300 steps. In Mozer and Bachrach (1989), we consider why the weight constraints are harmful and suggest several remedies. Without weight constraints, the resulting weight matrix, which contains a collection of positive and negative weights of varying magnitudes, is not readily interpreted. In the case of the $n$-room world, one reason why the final weights are difficult to interpret is because the net has discovered a solution that does not satisfy the RS update graph formalism; it has discovered the notion of complementation links of the sort shown in Figure 1d. With the use of complementation links, only three units are required, not six. Consequently, the three unnecessary units are either cut out of the solution or encode information redundantly.

Table 1 compares the performance of the RS algorithm against that of the connectionist network without weight constraints for several environments. Performance is measured in terms of the median number of actions the robot must take before it is able to predict the outcome of subsequent actions. (Further details of the experiments can be found in Mozer and Bachrach, 1989.) In simple environments, the connectionist update graph can outperform the RS algorithm. This result is quite surprising when considering that the action sequence used to train the network is generated at random, in contrast to the RS algorithm, which involves a strategy for exploring the environment. We conjecture that the network does as well as it does because it considers and updates many hypotheses in parallel at each time step. In complex environments, however, the network does poorly. By "complex", we mean that the number of nodes in the update graph is quite large and the number of distinguishing environmental sensations is relatively small. For example, the network failed to learn a 32-room world, whereas the RS algorithm succeeded. An intelligent exploration strategy seems necessary in this case: random actions will take too long to search the state space. This is one direction our future work will take.

Beyond the potential speedups offered by connectionist learning algorithms, the connectionist approach has other benefits.

Table 1: Number of Steps Required to Learn Update Graph

| Environment | RS Algorithm | Connectionist Update Graph |
|---|---|---|
| Little Prince World | 200 | 91 |
| Car Radio World | 27,695 | 8,167 |
| Four-Room World | 1,388 | 1,308 |
| 32-Room World | 52,436 | fails |

- Performance of the network appears insensitive to prior knowledge of the number of nodes in the update graph being learned. In contrast, the RS algorithm requires an upper bound on the update graph complexity, and performance degrades significantly if the upper bound isn't tight.

- The network is able to accommodate "noisy" environments, also in contrast to the RS algorithm.

- During learning, the network continually makes predictions about what sensations will result from a particular action, and these predictions improve with experience. The RS algorithm cannot make predictions until learning is complete; it could perhaps be modified to do so, but there would be an associated cost.

- Treating the update graph as matrices of connection strengths has suggested generalizations of the update graph formalism that don't arise from a more traditional analysis. First, there is the fairly direct extension of allowing complementation links. Second, because the connectionist network is a linear system, any rank-preserving linear transform of the weight matrices will produce an equivalent system, but one that does not have the local connectivity of the update graph (see Mozer & Bachrach, 1989). The linearity of the network also allows us to use tools of linear algebra to analyze the resulting connectivity matrices.

These benefits indicate that the connectionist approach to the environment-modeling problem is worthy of further study. We do not wish to claim that the connectionist approach supercedes the impressive work of Rivest and Schapire. However, it offers complementary strengths and alternative conceptualizations of the learning problem.

## Acknowledgements

Our thanks to Rob Schapire, Paul Smolensky, and Rich Sutton for helpful discussions. This work was supported by a grant from the James S. McDonnell Foundation to Michael Mozer, grant 87-2-36 from the Sloan Foundation to Geoffrey Hinton, and grant AFOSR-87-0030 from the Air Force Office of Scientific Research, Bolling AFB, to Andrew Barto.

## References

Mozer, M. C., & Bachrach, J. (1989). *Discovering the structure of a reactive environment by exploration* (Technical Report CU-CS-451-89). Boulder, CO: University of Colorado, Department of Computer Science.

Rivest, R. L., & Schapire, R. E. (1987). Diversity-based inference of finite automata. In *Proceedings of the Twenty-Eighth Annual Symposium on Foundations of Computer Science* (pp. 78-87).

Rivest, R. L., & Schapire, R. E. (1987). A new approach to unsupervised learning in deterministic environments. In P. Langley (Ed.), *Proceedings of the Fourth International Workshop on Machine Learning* (pp. 364-375).

Rumelhart, D. E., Hinton, G. E., & Williams, R. J. (1986). Learning internal representations by error propagation. In D. E. Rumelhart & J. L. McClelland (Eds.), *Parallel distributed processing: Explorations in the microstructure of cognition. Volume I: Foundations* (pp. 318-362). Cambridge, MA: MIT Press/Bradford Books.

Schapire, R. E. (1988). *Diversity-based inference of finite automata.* Unpublished master's thesis, Massachusetts Institute of Technology, Cambridge, MA.